# A Neurodynamical Approach to Visual Attention

**Gustavo Deco**
Siemens AG
Corporate Technology
Neural Computation, ZT IK 4
Otto-Hahn-Ring 6
81739 Munich, Germany
Gustavo.Deco@mchp.siemens.de

**Josef Zihl**
Institute of Psychology
Neuropsychology
Ludwig-Maximilians-University Munich
Leopoldstr. 13
80802 Munich, Germany

## Abstract

The psychophysical evidence for "selective attention" originates mainly from visual search experiments. In this work, we formulate a hierarchical system of interconnected modules consisting in populations of neurons for modeling the underlying mechanisms involved in selective visual attention. We demonstrate that our neural system for visual search works across the visual field in parallel but due to the different intrinsic dynamics can show the two experimentally observed modes of visual attention, namely: the serial and the parallel search mode. In other words, neither explicit model of a focus of attention nor saliencies maps are used. The focus of attention appears as an emergent property of the dynamic behavior of the system. The neural population dynamics are handled in the framework of the mean-field approximation. Consequently, the whole process can be expressed as a system of coupled differential equations.

## 1 Introduction

Traditional theories of human vision considers two functionally distinct stages of visual processing [1]. The first stage, termed *the preattentive stage*, implies an unlimited-capacity system capable of processing the information contained in the entire visual field in parallel. The second stage is termed *the attentive or focal stage*, and is characterized by the serial processing of visual information corresponding to local spatial regions. This stage of processing is typically associated with a limited-capacity system which allocates its resources to a single particular location in visual space. The designed psychophysical experiments for testing this hypothesis consist of visual search tasks. In a visual search test the subject have to look at the display containing a frame filled with randomly positioned items in order to seek for an a priori defined target item. All other items in a frame which are not the target are called distractors. The number of items in a frame is called the frame size. The relevant variable to be measured is the reaction time as a function of the frame size. In this context, the *Feature Integration Theory*, assumes that the two stage processes operate sequentially [1]. The first early preattentive stage runs in parallel over the complete visual field extracting single *primitive features* without

integrating them. The second attentive stage has been likened to a spotlight. This metaphor alludes that attention is focally allocated to a local region of the visual field where stimuli are processed in more detail and passed to higher level of processing, while, in the other regions not illuminated by the attentional spotlight, no further processing occurs. Computational models formulated in the framework of feature integration theory require the existence of a *saliency or priority map* for registering the potentially interesting areas of the retinal input, and a *gating* mechanism for reducing the amount of incoming visual information, so that limited computational resources in the system are not overloaded. The priority map serves to represent topographically the relevance of different parts of the visual field, in order to have a mechanism for guiding the attentional focus on salient regions of the retinal input. The focused area will be gated, such that only the information within will be passed further to yet higher levels, concerned with object recognition and action. The disparity between these two stages of attentional visual processing originated a vivid experimental disputation. Duncan and Humphreys [2] have postulated a hypothesis that integrates both attentional modes (parallel and serial) as an instantiates of a common principle. This principle sustains in both schemes that a selection is made. In the serial focal scheme, the selection acts on in the space dimension, while in the parallel spread scheme the selection concentrates in feature dimensions, e.g. color. On the other hand, Duncan's attentional theory [3] proposed that after a first parallel search a competition is initiated, which ends up by accepting only one object namely the target. Recently, several electrophysiological experiments have been performed which seems to support this hypothesis [4]. Chelazzi et al. [4] measured IT (inferotemporal) neurons in monkeys observing a display containing a target object (that the monkey has seen previously) and a distractor. They report a short period during which the neuron's response is enhanced. After this period the activity level of the neuron remains high if the target is the neuron's effective stimulus, and decay otherwise. The challenging question is therefore: is really the linear increasing reaction time observed in some visual search tests due to a serial mechanism? or is there only parallel processing followed by a dynamical time consuming latency? In other words, are really priority maps and spotlight paradigm required? or can a neurodynamical approach explain the observed psychophysical experiments?. Furthermore, it should be clarified if the feature dimension search is achieved independently in each feature dimension or is done after integrating the involved feature dimensions. We study in this paper these questions from a computational perspective. We formulate a neurodynamical model consisting in interconnected populations of biological neurons specially designed for visual search tasks. We demonstrate that it is plausible to build a neural system for visual search, which works across the visual field in parallel but due to the different intrinsic dynamics can show the two experimentally observed modes of visual attention, namely: the serial focal and the parallel spread over the space mode. In other words, neither explicit serial focal search nor saliency maps should be assumed. The focus of attention is not included in the system but just results after convergence of the dynamical behavior of the neural networks. The dynamics of the system can be interpreted as an intrinsic dynamical routing for binding features if top-down information is available. Our neurodynamical computational model requires independent competition mechanism along each feature dimension for explaining the experimental data, implying the necessity of the independent character of the search in separated and not integrated feature dimensions. The neural population dynamics are handled in the framework of the mean-field approximation yielding a system of coupled differential equations.

## 2 Neurodynamical model

We extend with the present model the approach of Usher and Niebur [5], which is based on the experimental data of Chelazzi et al. [4], for explaining the results of visual search experiments. The hierarchical architecture of our system is shown in Figure 1. The input retina is given as a matrix of visual items. The location of each item at the retina is

specified by two indices $ij$ meaning the position at the row $i$ and the column $j$. The dimension of this matrix is $SxS$, i.e. the frame size is also $SxS$. The information is processed at each spatial location in parallel. Different feature maps extract for the item at each position the local values of the features. In the present work we hypothesize that selective attention is guided by an independent mechanism which corresponds to the independent search of each feature. Let us assume that each visual item can be defined by $K$ features. Each feature $k$ can adopt $L(k)$ values, for example the feature color can have the values red or green (in this case $L(color)=2$). For each feature map $k$ exist $L(k)$ layers of neurons for characterizing the presence of each feature value.

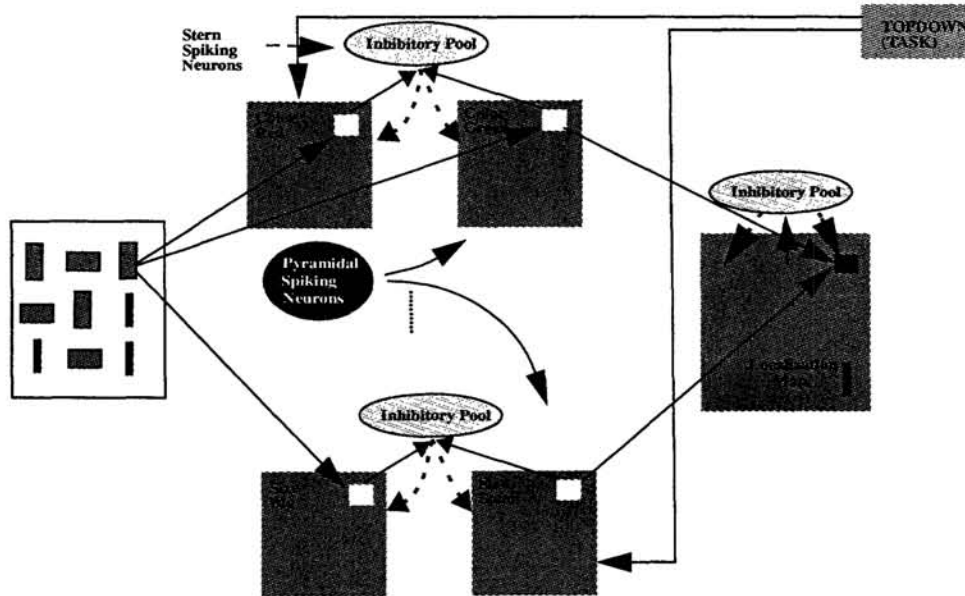

Figure 1: Hierarchical architecture of spiking neural modules for visual selective attention. Solid arrows denote excitatory connections and dotted arrows denote inhibitory connections

A cell assembly consisting in a population of full connected excitatory integrate-and-fire spiking neurons (pyramidal cells) is allocated in each layer and for each item location for encoding the presence of a specific feature value (e.g. color red) at the corresponding position. This corresponds to a sparse distributed representation. The feature maps are topographically ordered, i.e. the receptive fields of the neurons belonging to the cell assembly $ij$ at one of these maps are sensible to the location $ij$ at the retinal input. We further assume that the cell assemblies in layers corresponding to a feature dimension are mutually inhibitory. Inhibition is modeled, according to the constraint imposed by Dale's principle, by a different pool of inhibitory neurons. Each feature dimension has therefore an independent pool of inhibitory neurons. This accounts for the neurophysiological fact that the response of V4 neurons sensible to a specific feature value is enhanced and the activity of the other neurons sensible to other feature values are suppressed. A high level map consisting also in a topographically ordered excitatory cell assemblies is introduced for integration of the different feature dimension at each item location, i.e. for binding the features of each item. These cell assemblies are also mutually inhibited through a common

pool of inhibitory neurons. This layer corresponds to the modeling of IT neurons, which show location specific enhancement of activity by suppression of the responses of the cell assemblies associated to other locations. This fact would yield a dynamical formation of a focus of attention without explicitly assuming any spotlight. Top-down information consisting in the feature values at each feature dimension of the target item is feed in the system by including an extra excitatory input to the corresponding feature layers. The whole system analyzes the information at all locations in parallel. Larger reaction times correspond to slower dynamical convergence at all levels, i.e. feature map and integration map levels.

Instead of solving the explicit set of integrate-and-fire neural equations, the Hebbian cell assemblies adopted representation impels to adopt a dynamic theory whose dependent variables are the activation levels of the cell populations. Assuming an ergodic behavior [5] it is possible to derive the dynamic equations for the cell assembly activities level by utilizing the mean-field approximation [5]. The essential idea consists in characterizing each cell assembly by means of each activity $x$, and an input current that is characteristic for all cells in the population, denoted by $I$, which satisfies:

$$x = F(I) = \frac{1}{T_r - \tau \log\left(1 - \frac{1}{\tau I}\right)} \qquad (1)$$

which is the response function that transforms current into discharge rates for an integrate-and-fire spiking neuron with deterministic input, time membrane constant $\tau$ and absolute refractory time $T_r$. The system of differential equations describing the dynamics of the feature maps are:

$$\tau \frac{\partial}{\partial t} I_{ijkl}(t) = -I_{ijkl}(t) + aF(I_{ijkl}(t))$$

$$- bF(I^P_k(t)) + I_0 + I^F_{ijkl} + I^A_{kl} + \nu$$

$$\tau_P \frac{\partial}{\partial t} I^P_k(t) = -I^P_k(t) + c \sum_{i=1}^{S} \sum_{j=1}^{S} \sum_{k=1}^{L(k)} F(I_{ijkl}(t))$$

$$- dF(I^P_k(t))$$

where $I_{ijkl}(t)$ is the input current for the population with receptive field at location $ij$ of the feature map $k$ that analysis the value feature $l$, $I^P_k(t)$ is the current in the inhibitory pool bounded to the feature map layers of the feature dimension $k$. The frame size is $S$. The additive Gaussian noise $\nu$ considered has standard deviation 0.002. The synaptic time constants were $\tau = 5$ msec for the excitatory populations and $\tau_P = 20$ for the inhibitory pools. The synaptic weights chosen were: $a = 0.95, b = 0.8, c = 2.$ and $d = 0.1$. $I_0 = 0.025$ is a diffuse spontaneous background input, $I^F_{ijkl}$ is the sensory input to the cells in feature map $k$ sensible to the value $l$ and with receptive fields at the location $ij$ at the retina. This input characterizes the presence of the respective feature value at the corresponding position. A value of 0.05 corresponds to the presence of the respective feature value and a value of 0 to the absence of it. The top-down target information $I^A_{kl}$ was equal 0.005 for the layers which code the target properties and 0 otherwise.

The higher level integrating assemblies are described by following differential equation system:

$$\tau_H \frac{\partial}{\partial t} I^H{}_{ij}(t) = -I_{ij}(t) + \widehat{a}\, F(I_{ij}(t)) - \widehat{b}\, F(I^{PH}(t))$$

$$I_0 + \widehat{w} \sum_{k=1}^{K} \sum_{l=1}^{L(k)} F(I_{ijkl}(t)) + \nu$$

$$\tau_{PH} \frac{\partial}{\partial t} I^{PH}(t) = -I^{PH}(t) + \widehat{c} \sum_{i=1}^{S} \sum_{j=1}^{S} F(I^H{}_{ij}(t))$$

$$-\widehat{d}\, F(I^{PH}(t))$$

where $I^H{}_{ij}(t)$ is the input current for the population with receptive field at location $ij$ of the high level integrating map, $I^{PH}(t)$ is the associated current in the inhibitory pool. The synaptic time constants were $\tau_H = 5$ msec for the excitatory populations and $\tau_{PH} = 2C$ for the inhibitory pools. The synaptic weights chosen were: $\widehat{a} = 0.95$, $\widehat{b} = 0.8$, $\widehat{w} = 1$, $\widehat{c} = 1$. and $\widehat{d} = 0.1$.

These systems of differential equations were integrated numerically until a convergence criterion were reached. This criterion were that the neurons in the high level map are polarized, i.e.

$$F(I^H{}_{i_{max}j_{max}}(t)) - \frac{\displaystyle\sum_{i \ne i_{max}} \sum_{j \ne j_{max}} F(I^H{}_{ij}(t))}{(S^2 - 1)} > \theta$$

where the index $i_{max}j_{max}$ denotes the cell assembly in the high level map with maximal activity and the threshold $\theta$ was chosen equal to 0.1. The second in the l.h.s measure the mean distractor activity. At each feature dimension the fixed point of the dynamic is given by the activity of cell assemblies at the layers with a common value with the target and corresponding to items having this value. For example, if the target is red, at the color map, the activity at the green layer will be suppressed and the cell assemblies corresponding to red items will be enhanced. At the high-level map, the populations corresponding to location which are maximally in all feature dimensions activated will be enhanced by suppressing the others. In other words, the location that shows all feature dimension equivalent at what top-down is stimulated and required, will be enhanced when the target is at this location.

## 3 Simulations of visual search tasks

In this section we present results simulating the visual search experiments involving *feature* and *conjunction search* [1]. Let us define the different kinds of search tasks by given a pair of numbers $m$ and $n$, where $m$ is the number of feature dimensions by which the distractors differ from the target and $n$ is the number of feature dimensions by which the distractor groups simultaneously differ from the target. In other words, the feature search corresponds to a 1,1-search; a standard conjunction search corresponds to a 2,1-search; a triple conjunction search can be a 3,1 or a 3,2-search if the target differs from all distractor groups in one or in two features respectively. We assume that the items are defined by three feature dimensions ($K = 3$, e.g. color, size and position), each one

having two values ($L(k) = 2$ for $k = 1, 2, 3$). At each size we repeat the experiment 100 times, each time with different randomly generated distractors and target.

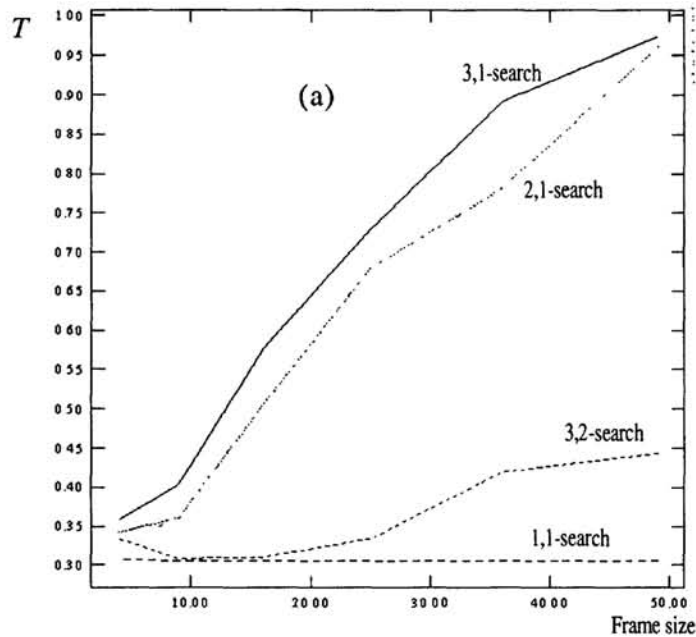

Figure 2: Search times for feature and conjunction searches obtained utilizing the presented model.

We plot as result the mean value $T$ of the 100 simulated reaction times (in msec) as a function of the frame size. In Figure 2, the results obtained for 1,1; 2,1; 3,1 and 3,2-searches are shown. The slopes of the reaction time vs. frame size curves for all simulations are absolutely consistent with the existing experimental results[1]. The experimental work reports that in feature search (1,1) the target is detected in parallel across the visual field. Furthermore, the slopes corresponding to standard conjunction search and triple conjunction search are a linear function of the frame size, where by the slope of the triple conjunction search is steeper or very flat than in the case of standard search (2,1) if the target differs from the distractors in one (3,1) or two features (3,2) respectively. In order to analyze more carefully the dynamical evolution of the system, we plot in Figure 3 the temporal evolution of the rate activity corresponding to the target and to the distractors at the high-level integrating map and also separately for each feature dimension level for a parallel (1,1-search) and a serial (3,1-search) visual tasks. The frame size used is 25. It is interesting to note that in the case of 1,1-search the convergence time in all levels are very small and therefore this kind of search appears as a parallel search. In the case of 3,1-search the latency of the dynamic takes more time and therefore this kind of search appears as a serial one, in spite that the underlying mechanisms are parallel. In this case (see Figure 3-c) the large competition present in each feature dimension delays the convergence of the dynamics at each feature dimension and therefore also at the high-level map. Note in Figure 3-c the slow suppression of the distractor activity that reflects the underlying competition.

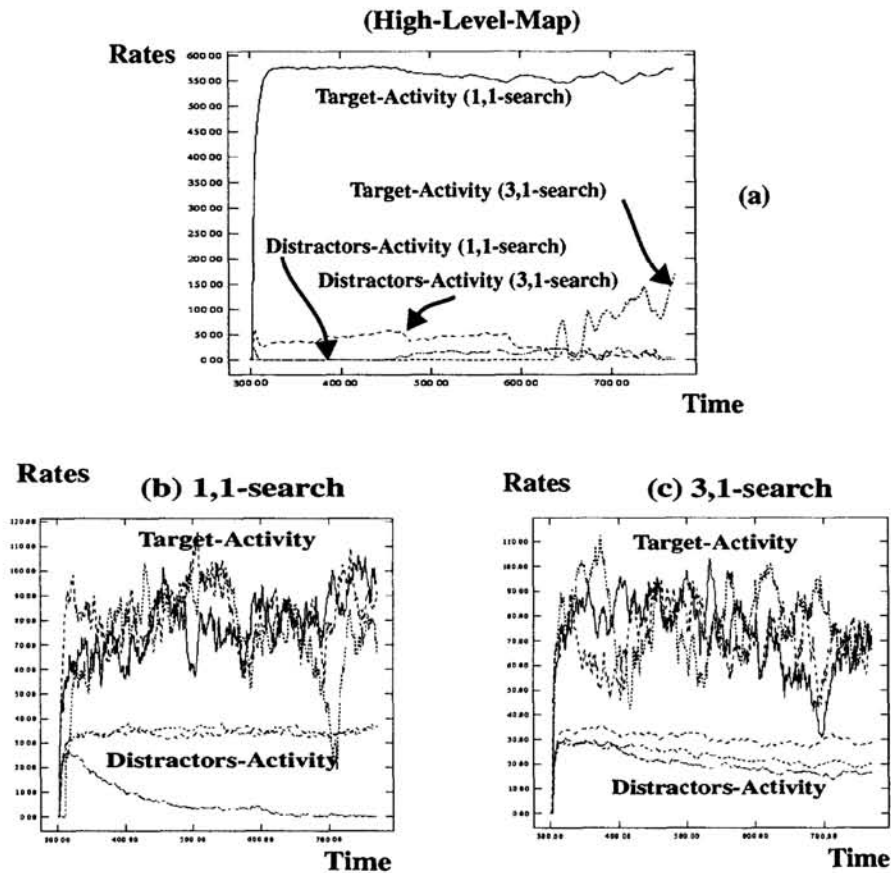

Figure 3: Activity levels during visual search experiments. (a) High-level-map rates for target $F(I^H_{i_{max}j_{max}}(t))$ and mean distractors-activity. (b) Feature-level map rates for target and one distractor activity for 1,1-search. There is one curve for each feature dimension (i.e. 3 for target and 3 for distractor. (c) the same as (b) but for 3,1-search.

# References

[1] Treisman, A. (1988) Features and objects: The fourteenth Barlett memorial lecture. *The Quarterly Journal of Experimental Psychology*, 40A, 201-237.

[2] Duncan, J. and Humphreys, G. (1989) Visual search and stimulus similarity. *Psychological Review*, **96**, 433-458.

[3] Duncan, J. (1980) The locus of interference in the perception of simultaneous stimuli. *Psychological Review*, **87**, 272-300.

[4] Chelazzi, L., Miller, E., Duncan, J. and Desimone, R. (1993) A neural basis for visual search in inferior temporal cortex. *Nature (London)*, **363**, 345-347.

[5] Usher, M. and Niebur, E. (1996) Modeling the temporal dynamics of IT neurons in visual search: A mechanism for top-down selective attention. *Journal of Cognitive Neuroscience*, **8**, 311-327.
